# Learning with Partially Absorbing Random Walks

**Xiao-Ming Wu**[1], **Zhenguo Li**[1], **Anthony Man-Cho So**[3], **John Wright**[1] and **Shih-Fu Chang**[1,2]

[1]Department of Electrical Engineering, Columbia University
[2]Department of Computer Science, Columbia University
[3]Department of SEEM, The Chinese University of Hong Kong
{xmwu, zgli, johnwright, sfchang}@ee.columbia.edu, manchoso@se.cuhk.edu.hk

## Abstract

We propose a novel stochastic process that is with probability $\alpha_i$ being absorbed at current state $i$, and with probability $1 - \alpha_i$ follows a random edge out of it. We analyze its properties and show its potential for exploring graph structures. We prove that under proper absorption rates, a random walk starting from a set $\mathcal{S}$ of low conductance will be mostly absorbed in $\mathcal{S}$. Moreover, the absorption probabilities vary slowly inside $\mathcal{S}$, while dropping sharply outside, thus implementing the desirable cluster assumption for graph-based learning. Remarkably, the partially absorbing process unifies many popular models arising in a variety of contexts, provides new insights into them, and makes it possible for transferring findings from one paradigm to another. Simulation results demonstrate its promising applications in retrieval and classification.

## 1 Introduction

Random walks have been widely used for graph-based learning, leading to a variety of models including PageRank [14] for web page ranking, hitting and commute times [8] for similarity measure between vertices, harmonic functions [20] for semi-supervised learning, diffusion maps [7] for dimensionality reduction, and normalized cuts [12] for clustering. In graph-based learning one often adopts the cluster assumption, which states that the semantics usually vary smoothly for vertices within regions of high density [17], and suggests to place the prediction boundary in regions of low density [5]. It is thus interesting to ask how the cluster assumption can be realized in terms of random walks.

Although a random walk appears to explore the graph globally, it converges to a stationary distribution determined solely by vertex degrees regardless of the starting points, a phenomenon well known as the mixing of random walks [11]. This causes some random walk approaches intended to capture non-local graph structures to fail, especially when the underlying graph is well connected, i.e., the random walk has a large mixing rate. For example, it was recently proven in [16] that under some mild conditions the hitting and commute times on large graphs do not take into account the global structure of the graph at all, despite the fact that they have integrated all the relevant paths on the graph. It is also shown in [13] that the "harmonic" walks [20] in high-dimensional spaces converge to a constant distribution as the data size approaches infinity, which is undesirable for classification and regression. These findings show that intuitions regarding random walks can sometimes be misleading, and should be taken with caution. A natural question is: can we design a random walk which implements the cluster assumption with some guarantees?

In this paper, we propose partially absorbing random walks (PARWs), a novel random walk model whose properties can be analyzed theoretically. In PARWs, a random walk is with probability $\alpha_i$ being absorbed at current state $i$, and with probability $1 - \alpha_i$ follows a random edge out of it. PARWs are guaranteed to implement the cluster assumption in the sense that under proper absorp-

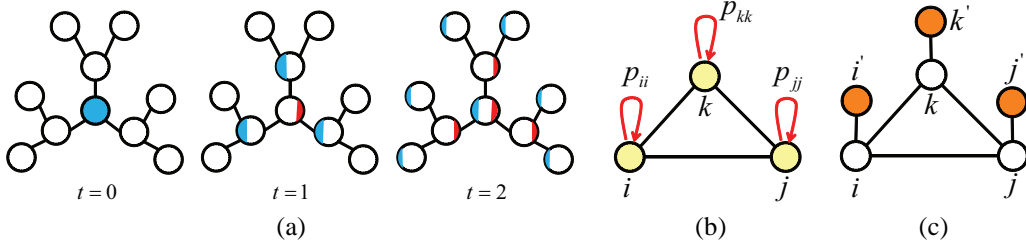

Figure 1: A partially absorbing random walk. (a) A flow perspective (see text). (b) A second-order Markov chain. (c) An equivalent standard Markov chain with additional sinks.

tion rates, a random walk starting from a set $\mathcal{S}$ of low conductance will be mostly absorbed in $\mathcal{S}$. Furthermore, we show that by setting the absorption rates, the absorption probabilities can vary s-lowly inside $\mathcal{S}$, while dropping sharply outside $\mathcal{S}$. This approximately piecewise constant property makes PARWs highly desirable and robust for a variety of learning tasks including ranking, clustering, and classification, as demonstrated in Section 4. More interestingly, it turns out that many existing models including PageRank, hitting and commute times, and label propagation algorithms in semi-supervised learning, can be unified or related in PARWs, which brings at least two benefits. On one hand, our theoretical analysis sheds some light on the understanding of existing models; on the other hand, it enables transferring findings among different paradigms. We present our model in Section 2, analyze a special case of it in Section 3, and show simulation results in Section 4. Section 5 concludes the paper. Most of our proofs are included in supplementary material.

## 2 Partially Absorbing Random Walks

Let us consider a simple diffusion process illustrated in Fig. 1(a). At the beginning, a unit flow (blue) is injected to the graph at a selected vertex. After one step, some of the flow (red) is "stored" at the vertex while the rest (blue) propagates to its neighbors. Whenever the flow passes a vertex, some fraction of it is retained at that vertex. As this process continues, the amount of flow stored in each vertex will accumulate and there will be less and less flow left running on the graph. After a certain number of steps, there will be almost no flow left running and the flow stored will nearly sum up to 1. The above diffusion process can be made precise in terms of random walks, as shown below.

Consider a discrete-time stochastic process $X = \{X_t : t \geq 0\}$ on the state space $N = \{1, 2, \ldots, n\}$, where the initial state $X_0$ is given, say $X_0 = i$, the next state $X_1$ is determined by the transition probability $\mathbb{P}(X_1 = j | X_0 = i) = p_{ij}$, and the subsequent states are determined by the transition probabilities

$$\mathbb{P}(X_{t+2} = j | X_{t+1} = i, X_t = k) = \left\{ \begin{array}{ll} 1, & i = j, i = k, \\ 0, & i \neq j, i = k, \\ \mathbb{P}(X_{t+2} = j | X_{t+1} = i) = p_{ij}, & i \neq k, \end{array} \right. \tag{1}$$

where $t \geq 0$. Note that the process $X$ is *time homogeneous*, i.e., the transition probabilities in (1) are independent of $t$. In other words, if the previous and current states are the same, the process will remain in the current state forever. Otherwise, the next state is conditionally independent of the previous state given the current state, i.e., the process behaves like a usual random walk.

To illustrate the above construction, consider Fig. 1(b). Starting from state $i$, there is some probability $p_{ii}$ that the process will stay at $i$ in the next step; and once it stays, the process will be absorbed into state $i$. Hence, we shall call the above process a *partially absorbing random walk* (PARW), where $p_{ii}$ is the *absorption rate* of state $i$. If $0 < p_{ii} < 1$, then we say that $i$ is a *partially absorbing state*. If $p_{ii} = 1$, then we say that $i$ is a *fully absorbing state*. Finally, if $p_{ii} = 0$, then we say that $i$ is a *transient state*. Note that if $p_{ii} \in \{0, 1\}$ for every state $i \in N$, then the above process reduces to a standard Markov chain [9].

A PARW is a second-order Markov chain completely specified by its *first order transition probabilities* $\{p_{ij}\}$. One can observe that any PARW can be realized as a standard Markov chain by adding a sink (fully absorbing state) to each vertex in the graph, as illustrated in Fig. 1(c). The transition

probability from $i$ to its sink $i'$ equals the absorption rate $p_{ii}$ in PARWs. One may also notice that the construction of PARWs can be generalized to the $m$-th order, i.e., the process is absorbed at a state only after it has stayed at that state for $m$-consecutive steps. However, it can be shown that any $m$-th order PARW can be realized by a second-order PARW. We will not elaborate on this due to space constraints.

## 2.1 PARWs on Graphs

Let $\mathcal{G} = (\mathcal{V}, W)$ be an undirected weighted graph, where $\mathcal{V}$ is a set of $n$ vertices and $W = [w_{ij}] \in \mathbb{R}^{n \times n}$ is a symmetric non-negative matrix of pairwise affinities among vertices. We assume $\mathcal{G}$ is connected. Let $D = \mathrm{diag}(d_1, d_2, \ldots, d_n)$ with $d_i = \sum_j w_{ij}$ as the degree of vertex $i$, and define the Laplacian of $\mathcal{G}$ by $L = D - W$ [6]. Denote by $d(\mathcal{S}) := \sum_{i \in S} d_i$ the volume of a subset $\mathcal{S} \subseteq \mathcal{V}$ of vertices. Let $\lambda_1, \lambda_2, \ldots, \lambda_n \geq 0$ be arbitrary, and set $\Lambda = \mathrm{diag}(\lambda_1, \lambda_2, \ldots, \lambda_n)$. Suppose that we define the first order transition probabilities of a PARW by

$$p_{ij} = \begin{cases} \frac{\lambda_i}{\lambda_i + d_i}, & i = j, \\ \frac{w_{ij}}{\lambda_i + d_i}, & i \neq j. \end{cases} \tag{2}$$

Then, we see that state $i$ is an absorbing state (either partially or fully) when $\lambda_i > 0$, and is a transient state when $\lambda_i = 0$. In particular, the matrix $\Lambda$ acts like a regularizer that controls the absorption rate of each state, i.e., the larger $\lambda_i$, the larger $p_{ii}$. In the sequel, we refer to $\Lambda$ as the *regularizer matrix*.

**Absorption Probabilities.** We are interested in the probability $a_{ij}$ that a random walk starting from state $i$, is absorbed at state $j$ in any finite number of steps. Let $A = [a_{ij}] \in \mathbb{R}^{n \times n}$ be the matrix of absorption probabilities. The following theorem shows that $A$ has a closed-form.

**Theorem 2.1.** *Suppose $\lambda_i > 0$ for some $i$. Then $A = (\Lambda + L)^{-1}\Lambda$.*

*Proof.* Since $\lambda_i > 0$ for some $i$, the matrix $\Lambda + L$ is positive definite and hence non-singular. Moreover, the matrix $\Lambda + D$ is non-singular, since $D$ is non-singular. Thus, the matrix $I - (\Lambda + D)^{-1}W = (\Lambda + D)^{-1}(\Lambda + L)$ is also non-singular. Now, observe that the absorbing probabilities $\{a_{ij}\}$ satisfy the following equations:

$$a_{ii} = \frac{\lambda_i}{\lambda_i + d_i} \times 1 + \sum_{j \neq i} \frac{w_{ij}}{\lambda_i + d_i} a_{ji}, \tag{3}$$

$$a_{ij} = \sum_{k \neq i} \frac{w_{ik}}{\lambda_i + d_i} a_{kj}, \ i \neq j. \tag{4}$$

Upon writing equations (3) and (4) in matrix form, we have $(I - (\Lambda + D)^{-1}W)A = (\Lambda + D)^{-1}\Lambda$, whence $A = (I - (\Lambda + D)^{-1}W)^{-1}(\Lambda + D)^{-1}\Lambda = (\Lambda + D - W)^{-1}\Lambda = (\Lambda + L)^{-1}\Lambda$. □

The following result confirms that $A$ is indeed a probability matrix.

**Proposition 2.1.** *Suppose $\lambda_i > 0$ for some $i$. Then $A$ is a non-negative matrix with each row summing up to 1.*

By Proposition 2.1, $\sum_k a_{jk} = 1$ for any $j$. This means that a PARW starting from any vertex will eventually be absorbed, provided that there is at least one absorbing state in the state space.

## 2.2 Limits of Absorption Probabilities

By Theorem 2.1, we see that the absorption probabilities ($A$) are governed by both the structure of the graph ($L$) and the regularizer matrix ($\Lambda$). It would be interesting to see how $A$ varies with $\Lambda$, particularly when $\lambda_i$'s become small which allows the flow to propagate sufficiently (Fig. 1(a)). The following result shows that as $\Lambda$ ($\lambda_i$'s) vanishes, each row of $A$ converges to a distribution proportional to $(\lambda_1, \lambda_2, \ldots, \lambda_n)$, regardless of graph structure.

**Theorem 2.2.** *Suppose $\lambda_i > 0$ for all $i$. Then*

$$\lim_{\alpha \to 0+} (\alpha \Lambda + L)^{-1} \alpha \Lambda = \mathbf{1}\bar{\boldsymbol{\lambda}}^{\top}, \tag{5}$$

*where $(\bar{\boldsymbol{\lambda}})_i = \lambda_i / (\sum_{j=1}^n \lambda_j)$. In particular, $\lim_{\alpha \to 0+} (\alpha I + L)^{-1} \alpha I = \frac{1}{n}\mathbf{1}\mathbf{1}^{\top}$.*

Theorem 2.2 tells us that with $\Lambda = \alpha I$ and as $\alpha \to 0$ a PARW will converge to the constant distribution $\mathbf{1}/n$, regardless of the starting vertex. At first glance, this limit seems meaningless. However, the following lemma will show that it actually has interesting connections with $L^+$, the pseudo-inverse of the graph Laplacian, a matrix that is widely studied and proven useful for many learning tasks including recommendation and clustering [8].

**Proposition 2.2.** *Suppose $\Lambda = \alpha I$ and denote $A^\alpha := (\Lambda + L)^{-1}\Lambda = (\alpha I + L)^{-1}\alpha$. Then,*

$$\lim_{\alpha \to 0} \frac{A^\alpha - \frac{1}{n}\mathbf{1}\mathbf{1}^\top}{\alpha} = L^+. \tag{6}$$

Proposition 2.2 gives a novel probabilistic interpretation of $L^+$. Note that by Theorem 2.2, $A^0 := \lim_{\alpha \to 0} A^\alpha = \frac{1}{n}\mathbf{1}\mathbf{1}^\top$. Thus $L^+$ is the derivative of $A^\alpha$ w.r.t. $\alpha$ at $\alpha = 0$, implying that $L^+$ reflects the variation of absorption probabilities when the absorption rate is very small. By (6), we see that ranking by $L^+$ is essentially the same as ranking by $A^\alpha$, when $\alpha$ is sufficiently small.

## 2.3 Relations with Popular Ranking and Classification Models

**Relations with PageRank Vectors.** Suppose $\lambda_j > 0$ for all $j$. Let $\mathbf{a}$ be the absorption probability vector of a PARW starting from vertex $i$. Denote by $\mathbf{s}$ the indicator vector of $i$, i.e., $\mathbf{s}(i) = 1$ and $\mathbf{s}(j) = 0$ for $j \neq i$. Then $\mathbf{a}^\top = \mathbf{s}^\top(\Lambda + L)^{-1}\Lambda$, which can be rewritten as

$$\mathbf{a}^\top = \mathbf{s}^\top(\Lambda + D)^{-1}\Lambda + \mathbf{a}^\top\Lambda^{-1}W(\Lambda + D)^{-1}\Lambda. \tag{7}$$

By letting $\Lambda = \frac{\beta}{1-\beta}D$, we have $\mathbf{a}^\top = \beta\mathbf{s}^\top + (1 - \beta)\mathbf{a}^\top D^{-1}W$, which is exactly the equilibrium equation for personalized PageRank [14]. Note that $\beta$ is often referred to as the "teleportation" probability in PageRank. This shows that personalized PageRank is a special case of PARWs with absorption rates $p_{ii} = \frac{\lambda_i}{\lambda_i + d_i} = \beta$.

**Relations with Hitting and Commute Times.** The hitting time $H_{ij}$ is the expected time that it takes a random walk starting from $i$ to first arrive at $j$, and the commute time $C_{ij}$ is the expected time it takes a random walk starting from $i$ to travel to $j$ and back to $i$, which can be computed as

$$H_{ij} = d(\mathcal{G})(L_{jj}^+ - L_{ij}^+), \quad C_{ij} = H_{ij} + H_{ji} = d(\mathcal{G})(L_{ii}^+ + L_{jj}^+ - 2L_{ij}^+), \tag{8}$$

where $d(\mathcal{G}) := \sum_i d_i$ denotes the volume of the graph. By (6), when $\Lambda = \alpha I$ and $\alpha$ is sufficiently small, ranking with $H_{ij}$ or $C_{ij}$ (say, with respect to $i$) is the same as ranking by $A_{jj}^\alpha - A_{ij}^\alpha$ or $A_{ii}^\alpha + A_{jj}^\alpha - 2A_{ij}^\alpha$ respectively. This appears to be not particularly meaningful because the term $A_{jj}^\alpha$ is the self-absorption probability that does not contain any essential information with the starting vertex $i$. Accordingly, it should not be included as part of the ranking function with respect to $i$. This argument is also supported in a recent study by [16], where the hitting and commute times are shown to be dominated by the inverse of degrees of vertices. In other words, they do not take into account the graph structure at all. A remedy they propose is to throw away the diagonal terms of $L^+$ and only use the off-diagonal terms. This happens to suggest using absorption probabilities for ranking and as similarity measure, because when $\alpha$ is sufficiently small, ranking by the off-diagonal terms of $L^+$ is essentially the same as ranking by $A_{ij}^\alpha$, i.e., the absorption probability of starting from $i$ and being absorbed at $j$. Our theoretical analysis in Section 3 and the simulation results in Section 4 further confirm this argument.

**Relations with Semi-supervised Learning.** Interestingly, many label propagation algorithms in semi-supervised learning can be cast in PARWs. The harmonic function method [20] is a PARW when setting $\lambda_i = \infty$ (absorption rate 1) for the labeled vertices while $\lambda_i = 0$ (absorption rate 0) for the unlabeled. In [19] the authors have made this interpretation in terms of absorbing random walks, where a random walk arriving at an absorbing state will stay there forever. PARWs can be viewed as an extension of absorbing random walks. The regularized harmonic function method [5] is also a PARW when setting $\lambda_i = \alpha$ for the labeled vertices while $\lambda_i = 0$ for the unlabeled. The consistency method [17], if using un-normalized Laplacian instead of normalized Laplacian, is a PARW with $\Lambda = \alpha I$. Our analysis in this paper reveals several nice properties of this case (Section 3). A variant of this method is a PARW with $\Lambda = \alpha D$, which is the same as PageRank as shown above. If we add an additional sink to the graph, a variant of harmonic function method [10] and a variant of the regularized harmonic function method [3] can all be included as instances of PARWs. We omit the details here due to space constraints.

**Benefits of a Unifying View.** We have shown that PARWs can unify or relate many models from different contexts. This brings at least two benefits. First, it sheds some light on existing models. For instance, hitting and commute times are not suitable for ranking given its interpretation in absorption probabilities, as discussed above. In the next section, we will show that a special case of PARWs is better suited for implementing the cluster assumption for graph-based learning. Second, a unifying view builds bridges between different paradigms thus making it easier to transfer findings between them. For example, it has been shown in [2, 4] that approximate personalized PageRank vectors can be computed in $O(1/\epsilon)$ iterations, where $\epsilon$ is a precision tolerance parameter. We indicate here that such a technique is also applicable to PARWs due to PARWs's generalizing nature. Consequently, most models included in PARWs can be substantially accelerated using the same technique.

## 3  PARWs with Graph Conductance

In this section, we present results on the properties of the absorption probability vector $\mathbf{a}_i$ obtained by a PARW starting from vertex $i$ (i.e., $\mathbf{a}_i^\top$ is the row $i$ of $A$). We show that properties of $\mathbf{a}_i$ relate closely to the connectivity between $i$ and the rest of graph, which can be captured by the conductance of the cluster $\mathcal{S}$ where $i$ belongs. We also find that properties of $\mathbf{a}_i$ depend on the setting of absorption rates. Our key results can be summarized as follows. In general, the probability mass of $\mathbf{a}_i$ is mostly absorbed by $\mathcal{S}$. Under proper absorption rates, $\mathbf{a}_i$ can vary slowly within $\mathcal{S}$ while dropping sharply outside $\mathcal{S}$. Such properties are highly desirable for learning tasks such as ranking, clustering, and classification.

The conductance of a subset $\mathcal{S} \subset \mathcal{V}$ of vertices is defined as $\Phi(\mathcal{S}) = \frac{w(\mathcal{S},\bar{\mathcal{S}})}{\min(d(\mathcal{S}),d(\bar{\mathcal{S}}))}$, where $w(\mathcal{S},\bar{\mathcal{S}}) := \sum_{(i,j)\in e(\mathcal{S},\bar{\mathcal{S}})} w_{ij}$ is the cut between $\mathcal{S}$ and its complement $\bar{\mathcal{S}}$ [6]. We denote the indicator vector of $\mathcal{S}$ by $\boldsymbol{\chi}_{\mathcal{S}}$ such that $\boldsymbol{\chi}_{\mathcal{S}}(i) = 1$ if $i \in \mathcal{S}$ and $\boldsymbol{\chi}_{\mathcal{S}}(i) = 0$ otherwise; and denote the stationary distribution w.r.t. $\mathcal{S}$ by $\boldsymbol{\pi}_{\mathcal{S}}$ such that $\boldsymbol{\pi}_{\mathcal{S}}(i) = d_i/d(\mathcal{S})$ if $i \in \mathcal{S}$ and $\boldsymbol{\pi}_{\mathcal{S}}(i) = 0$ otherwise. In terms of the conductance of $\mathcal{S}$, the following theorem gives an upper bound on the expected probability mass escaped from $\mathcal{S}$ if the distribution of the starting vertex is $\boldsymbol{\pi}_{\mathcal{S}}$.

**Theorem 3.1.** *Let $S$ be any set of vertices satisfying $d(S) \leq \frac{1}{2}d(G)$. Let $\gamma_1 = \min_{i\in\mathcal{S}} \frac{\lambda_i}{d_i}$ and $\gamma_2 = \max_{i\in\bar{\mathcal{S}}} \frac{\lambda_i}{d_i}$. Then,*

$$\boldsymbol{\pi}_{\mathcal{S}}^\top A \boldsymbol{\chi}_{\bar{\mathcal{S}}} \leq \frac{\gamma_2}{1+\gamma_2} \frac{1+\gamma_1}{\gamma_1^2} \Phi(\mathcal{S}). \tag{9}$$

Theorem 3.1 shows that most of the probability mass will be absorbed in $\mathcal{S}$, provided that $\mathcal{S}$ is of small conductance and the random walk starts from $\mathcal{S}$ according to $\boldsymbol{\pi}_{\mathcal{S}}$. In other words, a PARW will be trapped inside the cluster[1] from where it starts, as desired. To identify the entire cluster, what is more desirable would be that the absorption probabilities vary slowly within the cluster while dropping sharply outside. As such, the cluster can be identified by detecting the sharp drop. We show below that such property can be achieved by setting appropriate absorption rates at vertices.

### 3.1  PARWs with $\Lambda = \alpha I$

We will prove that the choice of $\Lambda = \alpha I$ can fulfill the above goal. Before presenting theoretical analysis, let us discuss the intuition behind it from both flow (Fig. 1(a)) and random walk perspectives. To vary slowly within the cluster, the flow needs to be distributed evenly within it; while to drop sharply outside, the flow must be prevented from escaping. This means that the absorption rates should be small in the interior but large near the boundary area of the cluster. Setting $\Lambda = \alpha I$ achieves this. It corresponds to the absorption rates $p_{ii} = \frac{\lambda_i}{\lambda_i+d_i} = \frac{\alpha}{\alpha+d_i}$, which decrease monotonically with $d_i$. Since the degrees of vertices are usually relatively large in the interior of the cluster due to denser connections, and small near its boundary area (Fig. 2(a)), the absorption rates are therefore much larger at its boundary than in its interior (Fig. 2(b)). State differently, a random walk may move freely inside the cluster, but it will get absorbed with high probability when traveling near the cluster's boundary. In this way, the absorption rates set up a bounding "wall" around the cluster to prevent the random walk from escaping, leading to an absorption probability vector that

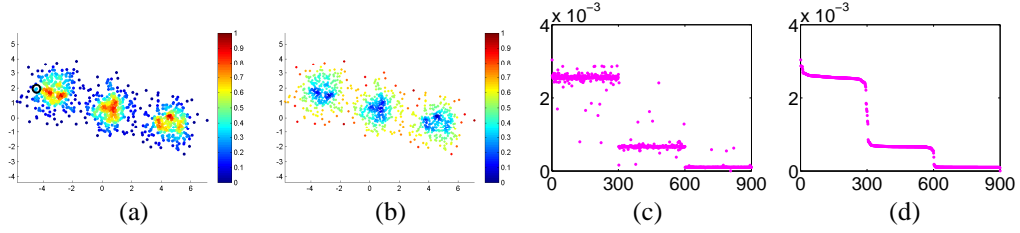

Figure 2: Absorption rates and absorption probabilities. (a) A data set of three Gaussians with the degrees of vertices in the underlying graph shown (see Section 4 for the descriptions of the data and graph construction). A starting vertex is denoted in black circle. (b–c) Absorption rates and absorption probabilities for $\Lambda = \alpha I$ ($\alpha = 10^{-3}$). (d) Sorted absorption probabilities of (c). For illustration purpose, in (a–b), the degrees of vertices and the absorption rates have been properly scaled, and in (c), the data are arranged such that points within each Gaussian appear consecutively.

varies slowly within the cluster while dropping sharply outside (Figs. 2(c–d)), thus implementing the cluster assumption. We make these arguments precise below.

It is worth pointing out that a PARW with $\Lambda = \alpha I$ is symmetric, i.e., the absorption probability of starting from $i$ and absorbed at $j$ is equal to the probability of starting from $j$ and absorbed at $i$. For simplicity, we use the abbreviated notation $\mathbf{a}$ to denote $\mathbf{a}_i$, the absorption probability vector for the PARW starting from vertex $i$. By (3) and the symmetry property, we immediately see that $\mathbf{a}$ has the following "harmonic" property:

$$\mathbf{a}(i) = \frac{\lambda_i}{\lambda_i + d_i} + \sum_{k \neq i} \frac{w_{ik}}{\lambda_i + d_i}\mathbf{a}(k), \quad \mathbf{a}(j) = \sum_{k \neq j} \frac{w_{jk}}{\lambda_j + d_j}\mathbf{a}(k), \quad j \neq i. \tag{10}$$

We will use this property to prove some interesting results. Another desirable property one should notice for this PARW is that the starting vertex always has the largest absorption probability, as shown by the following lemma.

**Lemma 3.2.** *Given $\Lambda = \alpha I$, then $a_{ii} > a_{ij}$ for any $i \neq j$.*

By Lemma 3.2 and without loss of generality, we assume the vertices are sorted so that $\mathbf{a}(1) > \mathbf{a}(2) \geq \cdots \geq \mathbf{a}(n)$, where vertex 1 is the starting vertex. Let $\mathcal{S}_k$ be the set of vertices $\{1, \ldots, k\}$. Denote $e(\mathcal{S}_i, \mathcal{S}_j)$ as the set of edges between $\mathcal{S}_i$ and $\mathcal{S}_j$.

The following theorem quantifies the drop of the absorption probabilities between $\mathcal{S}_k$ and $\bar{\mathcal{S}}_k$.

**Theorem 3.3.** *For every $\mathcal{S} \in \{\mathcal{S}_k \mid k = 1, 2, \ldots, n\}$,*

$$\sum_{(u,v) \in e(\mathcal{S}, \bar{\mathcal{S}})} w_{uv}\left(\mathbf{a}(u) - \mathbf{a}(v)\right) = \alpha\left(1 - \sum_{k \in \mathcal{S}} \mathbf{a}(k)\right). \tag{11}$$

Theorem 3.3 shows that the weighted difference in absorption probabilities between $\mathcal{S}_k$ and $\bar{\mathcal{S}}_k$ is $\alpha\left(1 - \sum_{j=1}^{k} \mathbf{a}(j)\right)$, implying that it drops slowly when $\alpha$ is small and as $k$ increases, as expected. Next we show the variation of absorption probabilities with graph conductance. Without loss of generality, we consider sets $\mathcal{S}_j$ where $d(\mathcal{S}_j) \leq \frac{1}{2}d(\mathcal{G})$.

The following theorem says that $\mathbf{a}(j+1)$ will drop little from $\mathbf{a}(j)$ if the set $\mathcal{S}_j$ has high conductance or if the vertex $j$ is far away from the starting vertex 1 (i.e., $j \gg 1$).

**Lemma 3.4.** *If $\Phi(\mathcal{S}_j) = \phi$, then*

$$\mathbf{a}(j + 1) \geq \mathbf{a}(j) - \frac{\alpha\left(1 - \sum_{k=1}^{j} \mathbf{a}(k)\right)}{\phi d(\mathcal{S}_j)}. \tag{12}$$

The above result can be extended to describe the drop in a much longer range, as stated in the following theorem.

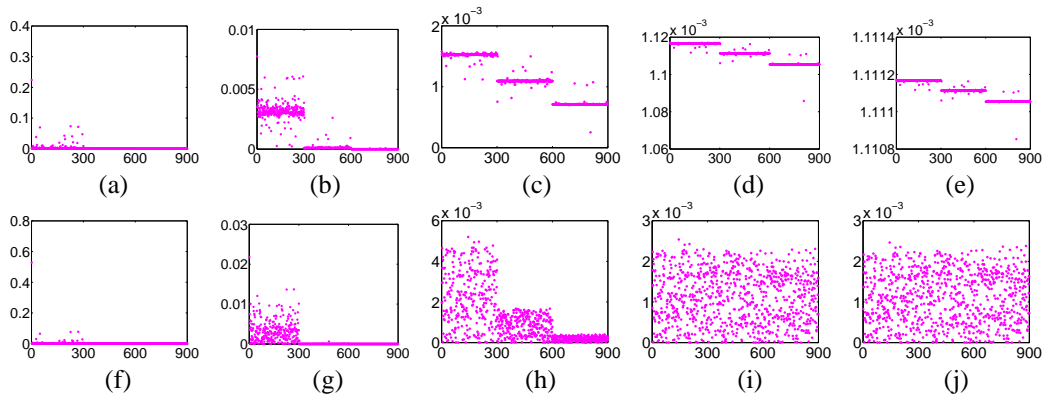

Figure 3: Absorption probabilities on the three Gaussians in Fig. 2(a) with the starting vertex denoted in black circle. (a–e) $\Lambda = \alpha I$, $\alpha = 10^0, 10^{-2}, 10^{-4}, 10^{-6}, 10^{-8}$; (f–j) $\Lambda = \alpha D$, $\alpha = 10^0, 10^{-2}, 10^{-4}, 10^{-6}, 10^{-8}$. For illustration purpose, the data are arranged such that points within each Gaussian appear consecutively, as in Fig. 2(c).

Table 1: Ranking results (MAP) on USPS

| Digits | 0 | 1 | 2 | 3 | 4 | 5 | 6 | 7 | 8 | 9 | All |
|---|---|---|---|---|---|---|---|---|---|---|---|
| $\Lambda = \alpha I$ | **.981** | **.988** | **.876** | **.893** | **.646** | **.778** | **.940** | **.919** | **.746** | **.730** | **.850** |
| PageRank | .886 | .972 | .608 | .764 | .488 | .568 | .837 | .825 | .626 | .702 | .728 |
| Manifold Ranking | .957 | .987 | .827 | .827 | .467 | .630 | .917 | .822 | .675 | .719 | .783 |
| Euclidean Distance | .640 | .980 | .318 | .499 | .337 | .294 | .548 | .620 | .368 | .480 | .508 |

**Theorem 3.5.** *If $\Phi(\mathcal{S}_j) \geq 2\phi$, then there exists a $k > j$ such that*

$$d(\mathcal{S}_k) \geq (1+\phi)d(\mathcal{S}_j) \quad and \quad \mathbf{a}(k) \geq \mathbf{a}(j) - \frac{\alpha \left(1 - \sum_{k=1}^{j} \mathbf{a}(k)\right)}{\phi d(\mathcal{S}_j)}.$$

Theorem 3.5 tells us that if the set $\mathcal{S}_j$ has high conductance, then there will be a set $\mathcal{S}_k$ much larger than $\mathcal{S}_j$ where the absorption probability $\mathbf{a}(k)$ remains large. In other words, $\mathbf{a}(k)$ will not drop much if $\mathcal{S}_j$ is closely connected with the rest of graph. Combining Theorems 3.3, 3.5, and 3.1, we see that the absorption probability vector of the PARW with $\Lambda = \alpha I$ has the nice property of varying slowly within the cluster while dropping sharply outside.

We remark that similar analyses have been conducted in [1, 2] on personalized PageRank, for the local clustering problem [15] whose goal is to find a local cut of low conductance near a specified starting vertex. As shown in Section 2, personalized PageRank is a special case of PARWs with $\Lambda = \alpha D = \frac{\beta}{1-\beta} D$, which corresponds to setting the same absorption rate $p_{ii} = \beta$ at each vertex. This setting does not take advantage of the cluster assumption. Indeed, despite the significant cluster structure in the three Gaussians (Fig. 2), no clear drop emerges by varying $\beta$ (Section 4). This explains the "heuristic" used in [1, 2] where the personalized PageRank vector is divided by the degrees of vertices to generate a sharp drop. In contrast, our choice of $\Lambda = \alpha I$ appears to be more justified, without the need of such post-processing while retaining a probabilistic foundation.

## 4 Simulation

In this section, we demonstrate our theoretical results on both synthetic and real data. For each data set, a weighted $k$-NN graph is constructed with $k = 20$. The similarity between vertices $i$ and $j$ is computed as $w_{ij} = \exp(-d_{ij}^2/\sigma)$ if $i$ is within $j$'s $k$ nearest neighbors or vice versa, and $w_{ij} = 0$ otherwise ($w_{ii} = 0$), where $\sigma = 0.2 \times r$ and $r$ denotes the average square distance between each point to its 20th nearest neighbor.

The first experiment is to examine the absorption probabilities when varying absorption rates. We use the synthetic three Gaussians in Fig. 2(a), which consists of 900 points from three Gaussians, with 300 in each. Fig. 3 compares the cases of $\Lambda = \alpha I$ and $\Lambda = \alpha D$ (PageRank). We can

Table 2: Classification accuracy on USPS

| HMN | LGC | $\Lambda = \alpha D$ | $\Lambda = \alpha I$ |
|---|---|---|---|
| $.782 \pm .068$ | $.792 \pm .062$ | $.787 \pm .048$ | $\mathbf{.881 \pm .039}$ |

draw several observations. For $\Lambda = \alpha I$, when $\alpha$ is large, most probability mass is absorbed in the cluster of the starting vertex (Fig. 3(a)). As it becomes appropriately small, the probability mass distributes evenly within the cluster, and a sharp drop emerges (Fig. 3(b)). As $\alpha \to 0$, the probability mass distributes more evenly within each cluster and also on the entire graph (Figs. 3(c–e)), but the drops between clusters are still quite significant. In contrast, for $\Lambda = \alpha D$, no significant drops show for all $\alpha$'s (Figs. 3(f–j)). This is due to the uniform absorption rates on the graph, which makes the flow favor vertices with denser connections (i.e., of large degrees). These observations support the theoretical arguments in Section 3 for PARWs with $\Lambda = \alpha I$ and suggest its robustness in distinguishing between different clusters.

The second experiment is to test the potential of PARWs for information retrieval. We compare PARWs with $\Lambda = \alpha I$ to PageRank (i.e., PARWs with $\Lambda = \alpha D$), Manifold Ranking [18], and the baseline using Euclidean distance. For parameter selection, we use $\alpha = 10^{-6}$ for $\Lambda = \alpha I$ and $\beta = 0.15$ for PageRank (see Section 2.3) as suggested in [14]. The regularization parameter in Manifold Ranking is set to 0.99, following [18]. The image benchmark USPS[2] is used for this experiment, which contains 9298 images of handwritten digits from 0 to 9 of size $16 \times 16$, with 1553, 1269, 929, 824, 852, 716, 834, 792, 708, and 821 instances of each digit respectively. Each instance is used as a query and the mean average precision (MAP) is reported. The results are shown in Table 1. We see that the PARW with $\Lambda = \alpha I$ consistently gives best results for individual digits as well as the entire data set.

In the last experiment, we test PARWs on classification/semi-supervised learning, also on USPS with all 9298 images. We randomly sample 20 instances as labeled data and make sure there is at least one label for each class. For PARWs, we classify each unlabeled instance $u$ to the class of the labeled vertex $v$ where $u$ is most likely to be absorbed, i.e., $v = \arg\max_{i \in \mathcal{L}} a_{ui}$ where $\mathcal{L}$ denotes the labeled data and $a_{ui}$ is the absorption probability. We compare PARWs with $\Lambda = \alpha I$ ($\alpha = 10^{-6}$) and $\Lambda = \alpha D$ ($\beta = 0.15$) to the harmonic function method (HMN) [20] coupled with class mass normalization (CMN) and the local and global consistency (LGC) method [17]. No parameter in HMN is required, and the regularization parameter in LGC is set to 0.99 following [17]. The classification accuracy averaged over 1000 runs is shown in Table 2. Again, it confirms the superior performance of the PARW with $\Lambda = \alpha I$.

In the second and third experiments, we also tried other parameter settings for methods where appropriate. We found that the performance of PARWs with $\Lambda = \alpha I$ is quite stable with small $\alpha$, and varying parameters in other methods did not lead to significantly better results, which validates our previous arguments.

## 5   Conclusions

We have presented partially absorbing random walks (PARWs), a novel stochastic process generalizing ordinary random walks. Surprisingly, it has been shown to unify or relate many popular existing models and provide new insights. Moreover, a new algorithm developed from PARWs has been theoretically shown to be able to reveal cluster structure under the cluster assumption. Simulation results have confirmed our theoretical analysis and suggested its potential for a variety of learning tasks including retrieval, clustering, and classification. In future work, we plan to apply our model to real applications.

## Acknowledgements

This work is supported in part by Office of Naval Research (ONR) grant #N00014-10-1-0242. The authors would like to thank the anonymous reviewers for their insightful comments.

## Footnotes

[1]A cluster is understood as a subset of vertices of small conductance.

[2]http://www-stat.stanford.edu/ tibs/ElemStatLearn/

# References

[1] R. Andersen and F. Chung. Detecting sharp drops in pagerank and a simplified local partitioning algorithm. *Theory and Applications of Models of Computation*, pages 1–12, 2007.

[2] R. Andersen, F. Chung, and K. Lang. Local graph partitioning using pagerank vectors. In *FOCS*, pages 475–486, 2006.

[3] Y. Bengio, O. Delalleau, and N. Le Roux. Label propagation and quadratic criterion. *Semi-supervised learning*, pages 193–216, 2006.

[4] P. Berkhin. Bookmark-coloring algorithm for personalized pagerank computing. *Internet Mathematics*, 3(1):41–62, 2006.

[5] O. Chapelle and A. Zien. Semi-supervised classification by low density separation. In *AIS-TATS*, 2005.

[6] F. Chung. *Spectral Graph Theory*. American Mathematical Society, 1997.

[7] R. Coifman and S. Lafon. Diffusion maps. *Applied and Computational Harmonic Analysis*, 21(1):5–30, 2006.

[8] F. Fouss, A. Pirotte, J. Renders, and M. Saerens. Random-walk computation of similarities between nodes of a graph with application to collaborative recommendation. *IEEE Transactions on Knowledge and Data Engineering*, 19(3):355–369, 2007.

[9] J. Kemeny and J. Snell. *Finite markov chains*. Springer, 1976.

[10] B. Kveton, M. Valko, A. Rahimi, and L. Huang. Semisupervised learning with max-margin graph cuts. In *AISTATS*, pages 421–428, 2010.

[11] L. Lovász and M. Simonovits. The mixing rate of markov chains, an isoperimetric inequality, and computing the volume. In *FOCS*, pages 346–354, 1990.

[12] M. Meila and J. Shi. A random walks view of spectral segmentation. In *AISTATS*, 2001.

[13] B. Nadler, N. Srebro, and X. Zhou. Statistical analysis of semi-supervised learning: The limit of infinite unlabelled data. In *NIPS*, pages 1330–1338, 2009.

[14] L. Page, S. Brin, R. Motwani, and T. Winograd. The pagerank citation ranking: Bringing order to the web. 1999.

[15] D. A. Spielman and S.-H. Teng. A local clustering algorithm for massive graphs and its application to nearly-linear time graph partitioning. *CoRR*, abs/0809.3232, 2008.

[16] U. Von Luxburg, A. Radl, and M. Hein. Hitting and commute times in large graphs are often misleading. *Arxiv preprint arXiv:1003.1266*, 2010.

[17] D. Zhou, O. Bousquet, T. Lal, J. Weston, and B. Schölkopf. Learning with local and global consistency. In *NIPS*, pages 595–602, 2004.

[18] D. Zhou, J. Weston, A. Gretton, O. Bousquet, and B. Schölkopf. Ranking on data manifolds. In *NIPS*, 2004.

[19] X. Zhu and Z. Ghahramani. Learning from labeled and unlabeled data with label propagation. Technical Report CMU-CALD-02-107, Carnegie Mellon University, 2002.

[20] X. Zhu, Z. Ghahramani, and J. Lafferty. Semi-supervised learning using gaussian fields and harmonic functions. In *ICML*, 2003.

